# Gene Expression Clustering with Functional Mixture Models

**Darya Chudova,**
Department of Computer Science
University of California, Irvine
Irvine CA 92697-3425
dchudova@ics.uci.edu

**Christopher Hart**
Division of Biology
California Institute of Technology
Pasadena, CA 91125
hart@caltech.edu

**Eric Mjolsness**
Department of Computer Science
University of California, Irvine
Irvine CA 92697-3425
emj@uci.edu

**Padhraic Smyth**
Department of Computer Science
University of California, Irvine
Irvine CA 92697-3425
smyth@ics.uci.edu

## Abstract

We propose a functional mixture model for simultaneous clustering and alignment of sets of curves measured on a discrete time grid. The model is specifically tailored to gene expression time course data. Each functional cluster center is a nonlinear combination of solutions of a simple linear differential equation that describes the change of individual mRNA levels when the synthesis and decay rates are constant. The mixture of continuous time parametric functional forms allows one to (a) account for the heterogeneity in the observed profiles, (b) align the profiles in time by estimating real-valued time shifts, (c) capture the synthesis and decay of mRNA in the course of an experiment, and (d) regularize noisy profiles by enforcing smoothness in the mean curves. We derive an EM algorithm for estimating the parameters of the model, and apply the proposed approach to the set of cycling genes in yeast. The experiments show consistent improvement in predictive power and within cluster variance compared to regular Gaussian mixtures.

## 1 Introduction

Curve data arises naturally in a variety of applications. Each curve typically consists of a sequence of measurements as a function of discrete time or some other independent variable. Examples of such data include trajectory tracks of individuals or objects (Gaffney and Smyth, 2003) and biomedical measurements of response to drug therapies (James and Sugar, 2003). In some cases, the curve data is measured on regular grids and the curves have the same lengths. It is straightforward to treat such curves as elements of the corresponding vector spaces, and apply traditional vector based clustering methodologies such as k-means or mixtures of Gaussian distributions Often the curves are sampled irregularly, have varying lengths, lack proper alignment in the time domain or the task requires interpolation

or inference at the off-grid locations. Such properties make vector-space representations undesirable. Curve data analysis is typically referred to as "functional data analysis" in the statistical literature (Ramsay and Silverman, 1997), where the observed measurements are treated as samples from an assumed underlying continuous-time process. Clustering in this context can be performed using mixtures of continuous functions such as splines (James and Sugar, 2003) and polynomial regression models (DeSarbo and Cron, 1988; Gaffney and Smyth, 2003). In this paper we focus on the specific problem of analyzing gene expression time course data and extend the functional mixture modelling approach to (a) cluster the data using plausible biological models for the expression dynamics, and (b) align the expression profiles along the time axis.

Large scale gene expression profiling measures the relative abundance of tens of thousands of mRNA molecules in the cell simultaneously. The goal of clustering in this context is to discover groups of genes with similar dynamics and find sets of genes that participate in the same regulatory mechanism. For the most part, clustering approaches to gene expression data treat the observed curves as elements of the corresponding vector-space. A variety of vector-based clustering algorithms have been successfully applied, ranging from hierarchical clustering (Eisen et al., 1998) to model based methods (Yeung et al., 2001). However, approaches operating in the observed "gridded" domain of discrete time are insensitive to many of the constraints that the temporal nature of the data imposes, including

**Continuity of the temporal process:** The continuous-time nature of gene expression dynamics are quite important from a scientific viewpoint. There has been some previous work on continuous time models in this context, e.g., mixed effects mixtures of splines (Bar-Joseph et al., 2002) were applied to clustering and alignment of the cell-cycle regulated genes in yeast and good interpolation properties were demonstrated. However, such spline models are "black boxes" that can approximate virtually any temporal behavior — they do not take the specifics of gene regulation mechanisms into account. In contrast, in this paper we propose specific functional forms that are targeted at short time courses, in which fairly simple reaction kinetics can describe the possible dynamics.

**Alignment:** Individual genes within clusters of co-regulated genes can exhibit variations in the time of the onset of their characteristic behaviors or in their initial concentrations. Such differences can significantly increase within-cluster variability and produce incorrect cluster assignments. We address this problem by explicitly modelling the unknown real-valued time shifts between different genes.

**Smoothing.** The high noise levels of observed gene expression data imply the need for robust estimation of mean behavior. Functional models (such as those that we propose here) naturally impose smoothness in the learned mean curves, providing implicit regularization for such data.

While some of these problems have been previously addressed individually, no prior work handles all of them in a unified manner. The primary contributions of this paper are (a) a new probabilistic model based on functional mixtures that can simultaneously cluster and align sets of curves observed on irregular time grids, and (b) a proposal for a specific functional form that models changes in mRNA levels for short gene expression time courses.

## 2 Model Description

### 2.1 Generative Model

We describe a generative model that allows one to simulate heterogeneous sets of curves from a mixture of functional curve models. Each generated curve $\mathbf{Y}_i$ is a series of obser-

vations at a discrete set of values $\mathbf{X}_i$ of an independent variable. In many applications, and for gene expression measurements in particular, the independent variable $\mathbf{X}$ is time.

We adopt the same general approach to functional curve clustering that is used in regression mixture models (DeSarbo and Cron, 1988), random effects regression mixtures (Gaffney and Smyth, 2003) and mixtures of spline models (James and Sugar, 2003). In all of these models, the component densities are conditioned on the values of the independent variable $\mathbf{X}_i$, and the conditional likelihood of a set $\mathbf{Y}$ of $N$ curves is defined as

$$P(\mathbf{Y}|\mathbf{X}, \Theta) = \prod_{i=1}^{N} \sum_{k=1}^{K} P(\mathbf{Y}_i|\mathbf{X}_i, \Theta_k)P(k) \tag{1}$$

Here $P(k)$ is the component probability and $\Theta$ is a complete set of model parameters. The clusters are defined by their mean curves parametrized by a set of parameters $\boldsymbol{\mu}_k$: $f_k(x) = f(x, \boldsymbol{\mu}_k)$, and a noise model that describes the deviation from the mean functional form (described below in Section 2.2.

In contrast to standard Gaussian mixtures, the functional mixture is defined in continuous time, allowing evaluation of the mean curves on a continuum of "off-grid" time points. This allows us to extend the functional mixture models described above by incorporating real-valued alignment of observed curves along the time axis. In particular, the precise time grid $\mathbf{X}_i$ of observation $i$ is assumed unknown and is allowed to vary from curve to curve. This is common in practice when the measurement process cannot be synchronized from curve to curve. For simplicity we assume (unknown) linear shifts of the curves along the time axis. We fix the basic time grid $\mathbf{X}$, but generate each curve on its own grid $(\mathbf{X} + \phi_i)$ with a curve-specific time offset $\phi_i$. We treat the offset corresponding to curve $\mathbf{Y}_i$ as an additional real-valued latent variable in the model. The conditional likelihood of a single curve under component $k$ is calculated by integrating out all possible offset values:

$$P\left(\mathbf{Y}_i|\mathbf{X}, \Theta_k\right) = \int_{\phi_i} P(\mathbf{Y}_i|\mathbf{X} + \phi_i, \Theta_k)P(\phi_i|\Theta_k)d\phi_i \tag{2}$$

Finally, we assume that the measurements have additive Gaussian noise with zero mean and diagonal covariance matrix $\mathbf{C}_k$, and express the conditional likelihood as

$$P(\mathbf{Y}_i|\mathbf{X} + \phi_i, \Theta_k) \propto \mathbf{N}\left(\mathbf{Y}_i|f_k(\mathbf{X} + \phi_i), \mathbf{C}_k\right) \tag{3}$$

The full set of cluster parameters $\Theta_k$ includes mean curve parameters $\boldsymbol{\mu}_k$ that define $f_k(x)$, covariance matrix $\mathbf{C}_k$, cluster probability $P(k)$, and time shift probability $P(\phi|k)$: $\Theta_k = \{\boldsymbol{\mu}_k, \mathbf{C}_k, P(k), P(\phi|k)\}$

## 2.2 Functional form of the mean curves

The generative model described above uses a generic functional form $f(x, \boldsymbol{\mu})$ for the mean curves. In this section, we introduce a parametric representation of $f(x, \boldsymbol{\mu})$ that is specifically tailored to short gene expression time courses.

To a first-order approximation, the raw mRNA levels $\{v_1, \ldots, v_N\}$ measured in gene expression experiments can be modeled via a system of differential equations with the following structure (see Gibson and Mjolsness , eq. 1.19, and Mestl, Lemay, and Glass (1996)):

$$\frac{dv_i}{dt} = \sigma g_{1,i}(v_1, \ldots, v_N) - \varrho v_i g_{2,i}(v_1, \ldots, v_N) \tag{4}$$

The first term on the right hand side is responsible for the synthesis of $v_i$ with maximal rate $\sigma$, and the second term represents decay with maximal fractional rate $\varrho$. In general, we don't know the specific coefficients or nonlinear saturating functions $g_1$ and $g_2$ that define the right hand-side of the equations. Instead, we make a few simplifying assumptions about the equation and use it as a motivation for the parametric functional form that we propose below. Specifically, suppose that

- the set of $N$ heterogeneous variables can be divided into $K$ groups of variables, whose production is driven by similar mechanisms;
- the synthesis and decay functions $g_1$ and $g_2$ are approximately piecewise constant in time for any given group;
- there are at most two regimes involved in the production of $v_i$, each characterized by their own synthesis and decay rates — this is appropriate for short time courses;
- for each group there can be an unknown change point on the time axis where a relatively rapid switching between the two different regimes takes place, due to exogenous changes in the variables $(v_1, \ldots, v_N)$ outside the group.

Within the regions of constant synthesis and decay functions $g_1$ and $g_2$, we can solve equation (4) analytically and obtain a family of simple exponential solutions parametrized by $\boldsymbol{\mu}_1 = \{\nu, \sigma, \varrho\}$:

$$f^a(x, \boldsymbol{\mu}_1) = \left(\nu - \frac{\sigma}{\varrho}\right)e^{-\sigma x} + \frac{\sigma}{\varrho}, \tag{5}$$

This motivates us to construct the functional forms for the mean curves by concatenating two parameterized exponents, with an unknown change point and a smooth transition mechanism:

$$f(x, \boldsymbol{\mu}) = f^a(x, \boldsymbol{\mu}_1)\left(1 - \Phi(x, \delta, \psi)\right) + f^a(x, \boldsymbol{\mu}_2)\Phi(x, \delta, \psi) \tag{6}$$

Here $f^a(x, \boldsymbol{\mu}_1)$ and $f^a(x, \boldsymbol{\mu}_2)$ represent the exponents to the left and right of the switching point, with different sets of initial conditions, synthesis and decay rates denoted by parameters $\boldsymbol{\mu}_1$ and $\boldsymbol{\mu}_2$. The nonlinear sigmoid transfer function $\Phi(x, \delta, \psi)$ allows us to model switching between the two regimes at $x = \psi$ with slope $\delta$: $\Phi(x, \delta, \psi) = \left(1 + e^{-\delta(x-\psi)}\right)^{-1}$

The random effects on the time grid allow us to time-shift each curve individually by replacing $x$ with $(x + \phi_i)$ in Equation (6). There are other biologically plausible transformation on the curves in a cluster that we do not pursue in this paper, such as allowing $\psi$ to vary with each curve, or representing minor differences in the regulatory functions $g_{1,i}$ and $g_{2,i}$ which affect the timing of their transitions.

When learning these models from data, we restrict the class of functions in Equation (6) to those with non-negative initial conditions, synthesis and decay rates, as well as enforcing continuity of the exponents at the switching point: $f^a(\psi, \boldsymbol{\mu}_1) = f^a(\psi, \boldsymbol{\mu}_2)$. Finally, given that the log-normal noise model is well-suited to gene expression data (Yeung et al., 2001) we use the logarithm of the functional forms proposed in Equation (6) as a general class of functions that describe the mean behavior within the clusters.

## 3 Parameter Estimation

We use the well-known Expectation Maximization (EM) algorithm to simultaneously recover the full set of model parameters $\Theta = \{\Theta_1, \ldots, \Theta_K\}$, as well as the posterior joint

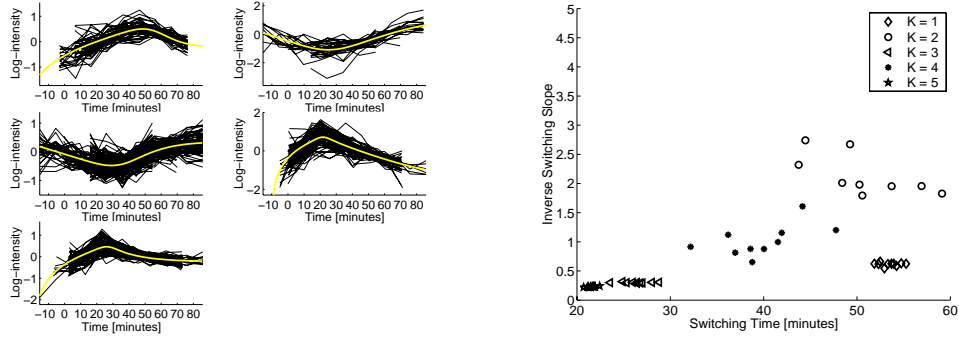

Figure 1: A view of the cluster mean curves (left) and variation in the switching-point parameters across 10 cross-validation folds (right) using functional clustering with alignment (see Section 4 for full details).

distribution of cluster membership $Z$ and time offsets $\phi$ for each observed curve. Each cluster is characterized by the parameters of the mean curves, noise variance, cluster probability and time shift distribution: $\Theta_k = \{\boldsymbol{\mu}_k, \mathbf{C}_k, P(k), P(\phi|k)\}$.

- In the E-step, we find the posterior distribution of the cluster membership $Z_i$ and the time shift $\phi_i$ for each curve $\mathbf{Y}_i$, given current cluster parameters $\Theta$;

- In the M-step, we maximize the expected log-likelihood with respect to the posterior distribution of $Z$ and $\phi$ by adjusting $\Theta$.

Since the time shifts $\phi$ are real-valued, the E-step requires evaluation of the posterior distribution over a continuous domain of $\phi$. Similarly, the M-step requires integration with respect to $\phi$. We approximate the domain of $\phi$ with a finite sample from its prior distribution. The sample is kept fixed throughout the computation. The posterior probability of the sampled values is updated after each M-step to approximate the model distribution $P(\phi|k)$.

The M-step optimization problem does not allow closed-form solutions due to non-linearities with respect to function parameters. We use conjugate gradient descent with a pseudo-Newton step size selection. The step size selection issue is crucial in this problem, as the second derivatives with respect to different parameters of the model differ by orders of magnitude. This indicates the presence of ridges and ravines on the likelihood surface, which makes gradient descent highly sensitive to the step size and slow to converge. To speed up the EM algorithm, we initialize the coefficients of the mean functional forms by approximating the mean vectors obtained using a standard vector-based Gaussian mixture model on the same data. This typically produces a useful set of initial parameter values which are then optimized by running the full EM algorithm for a functional mixture model with alignment.

We use the EM algorithm in its maximum a posteriori (MAP) formulation, using a zero-mean Gaussian prior distribution on the curve-specific time shifts. The variance of the prior distribution allows us to control the amount of shifting allowed in the model. We also use conjugate prior distributions for the noise variance $\mathbf{C}_k$ to regularize the model and prohibit degenerate solutions with near-zero covariance terms.

Figure 1 shows examples of mean curves (Equation (6)), that were learned from actual gene expression data. Each functional form has 7 free parameters: $\boldsymbol{\mu} = \{\nu, \sigma_1, \varrho_1, \sigma_2, \varrho_2, \delta, \psi\}$. Note that, as with many time-course gene expression data sets, having so few points presents an obvious problem for parameter estimation directly from a single curve. However, the curve-specific time shifts in effect provide a finer sampling grid that helps to

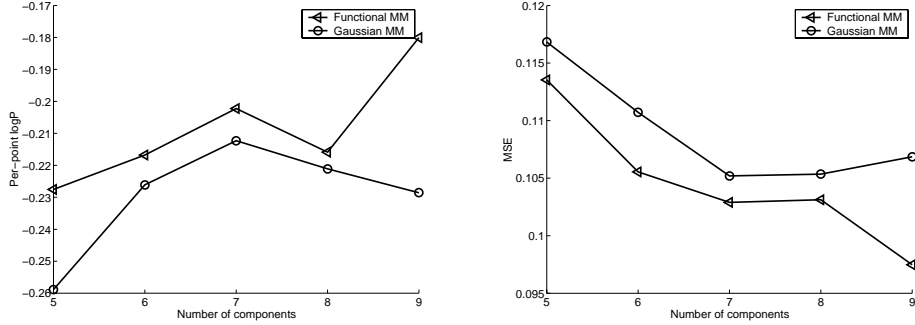

Figure 2: Cross-validated conditional logP scores (left) and cross-validated interpolation mean-squared error (MSE) (right), as a function of the number of mixture components, for the first cell cycle of the Cho et al. data set.

recover the parameters from observed data, in addition to the "pooling" effect of learning common functional forms for groups of curves. The right-hand side of Figure 1 shows a scatter plot of the switching parameters for 5 clusters estimated from 10 different cross-validation runs. The 5 clusters exhibit different dynamics (as indicated by the spread in parameter space) and the algorithm finds qualitatively similar parameter estimates for each cluster across the 10 different runs.

## 4 Experimental Evaluation

### 4.1 Gene expression data

We illustrate our approach using the immediate responses of yeast *Saccharomyces cerevisiae* when released from cell cycle arrest, using the raw data reported by Cho et al (1998). Briefly, the CDC28 TS mutants were released from the cell cycle arrest by temperature shift. Cells were harvested and RNA was collected every 10 min for 170 min, spanning two cell cycles. The RNA was than analyzed using Affymetrix gene chip arrays. From these data we select only the 416 genes which are reported to be actively regulated throughout the cell cycle and are expressed for 30 continuous minutes above an Affymetrix absolute level of 100 (a total of 385 genes pass these criteria). We normalize each gene expression vector by its median expression value throughout the time course to reduce the influence of probe-specific intensity biases.

### 4.2 Experimental results

In order to study the immediate cellular response we analyze only the first 8 time points of this data set. We evaluate the cross-validated out-of-sample performance of the proposed functional mixture model. A conventional Gaussian mixture model applied to observations on the discrete time grid is used for baseline comparison. It is not at all clear *a priori* that the functional mixture models with highly constrained parametric set of mean curves should outperform Gaussian mixtures that impose no parametric assumptions and are free to approximate any discrete grid observation. While one can expect that mixtures of splines (Bar-Joseph et al., 2002) or functions with universal approximation capabilities can be fitted to any mean behavior, the restricted class of functions that we proposed (based on the simplified dynamics of the mRNA changes implied by the differential equation in Equation (4)) is likely to fail if the true dynamics does not match the assumptions.

There are two main reasons to use the proposed restricted class of functional forms: (1)

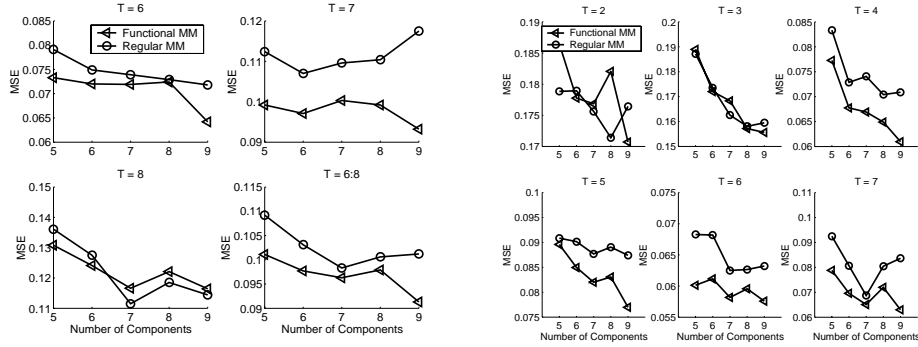

Figure 3: Cross-validated one-step-ahead prediction MSE (left) and cross-validated interpolation MSE (right) for the first cell cycle of the Cho et al. data set.

to be able to interpret the resulting mean curves in terms of the synthesis / decay rates at each of the regimes as well as the switching times; (2) to naturally incorporate alignment by real-values shifts along the time axis.

In Figures 2 and 3, we present 5-fold cross-validated out-of-sample scores, as a function of the number of clusters, for both the functional mixture model and the baseline Gaussian mixture model. The conditional logP score (Figure 2, left panel) estimates the average probability assigned to a single measurement at time points $6, 7, 8$ within an unseen curve, given the first five measurements of the same curve. Higher scores indicate a better fit. The conditioning on the first few time points allows us to demonstrate the power of models with random effects since estimation of alignment based on partial curves improves the probability of the remainder of the curve.

The interpolation error in Figure 2 (right panel) shows the accuracy of recovering missing measurements. The observed improvement in this score is likely due to the effect of aligning the test curves. To evaluate the interpolation error, we trained the models on the full training curves, and then assumed a single measurement was missing from the test curve (at time point 2 through 7). The model was then used to make a prediction at the time point of the missing measurement, and the interpolation error was averaged for all time points and test curves. The right panel of Figure 3 contains a detailed view of these results: each subplot shows the mean error in recovering values at a particular time point. While some time points are harder to approximate than the others (in particular, $T = 2, 3$), the functional mixture models provide better interpolation properties overall. Difficulties in approximating at $T = 2, 3$ can be attributed to the large changes in the intensities at these time points, and possibly indicate the limitations of the functional forms chosen as candidate mean curves.

Finally, the left panel of Figure 3 shows improvement in one-step-ahead prediction error. Again, we trained the models on the full curves, and then used the models to make prediction for test curves at time $T$ given all measurements up to $T - 1$ ($T = 6, 7, 8$). Figures 2 and 3 demonstrate a consistent improvement in the out-of-sample performance of the functional mixtures.

The improvements seen in these plots result from integrating alignment along the time axis into the clustering framework. We found that the functional mixture model without alignment does not result in better out-of-sample performance than discrete-time Gaussian mixtures. This may not be surprising given the constrained nature of the fitted functions.

In the experiments presented in this paper we used a Gaussian prior distribution on the time-

shift parameter to softly constrain the shifts to lie roughly within $1.5$ time grid intervals. The discrete grid alignment approaches that we proposed earlier in Chudova et al (2003) can successfully align curves if one assumes offsets on the scale of multiple time grid points. However, they are not designed to handle finer sub-grid alignments. Also worth noting is the fact that continuous time mixtures can align curves sampled on non-uniform time grids (such non-uniform sampling in time is relatively common in gene expression time course data).

## 5  Conclusions

We presented a probabilistic framework for joint clustering and alignment of gene expression time course data using continuous time cluster models. These models allow (1) real-valued off-grid alignment of unequally spaced measurements, (2) off-grid interpolation, and (3) regularization by enforcing smoothness implied by the functional cluster forms. We have demonstrated that a mixture of simple parametric functions with nonlinear transition between two exponential regimes can model a broad class of gene expression profiles in a single cell cycle of yeast. Cross-validated performance scores show the advantages of continuous time models over standard Gaussian mixtures. Possible extensions include adding additional curve-specific parameters, incorporating other alignment methods, and introducing periodic functional forms for multi-cycle data.

## References

Bar-Joseph, Z., Gerber, G., Gifford, D., Jaakkola, T., and Simon, I. (2002). A new approach to analyzing gene expression time series data. In *The Sixth Annual International Conference on (Research in) Computational (Molecular) Biology (RECOMB)*, pages 39–48, N.Y. ACM Press.

Cho, R. J., Campbell, M. J., Winzeler, E. A., Steinmetz, L., Conway, A., Wodicka, L., Wolfsberg, T. G., Gabrielian, A. E., Landsman, D., Lockhart, D. J., and Davis, R. W. (1998). A genome-wide transcriptional analysis of the mitotic cell cycle. *Mol Cell*, 2(1):65–73.

Chudova, D., Gaffney, S., Mjolsness, E., and Smyth, P. (2003). Mixture models for translation-invariant clustering of sets of multi-dimensional curves. In *Proceedings of the Ninth ACM SIGKDD International Conference on Knowledge Discovery and Data Mining*, pages 79–88, Washington, DC.

DeSarbo, W. S. and Cron, W. L. (1988). A maximum likelihood methodology for clusterwise linear regression. *Journal of Classification*, 5(1):249–282.

Eisen, M. B., Spellman, P. T., Brown, P. O., and Botstein, D. (1998). Cluster analysis and display of genome-wide expression patterns. *Proc Natl Acad Sci U S A*, 95(25):14863–8.

Gaffney, S. J. and Smyth, P. (2003). Curve clustering with random effects regression mixtures. In Bishop, C. M. and Frey, B. J., editors, *Proceedings of the Ninth International Workshop on Artificial Intelligence and Statistics*, Key West, FL.

Gibson, M. and Mjolsness, E. (2001). Modeling the activity of single genes. In Bower, J. M. and Bolouri, H., editors, *Computational Methods in Molecular Biology*. MIT Press.

James, G. M. and Sugar, C. A. (2003). Clustering for sparsely sampled functional data. *Journal of the American Statistical Association*, 98:397–408.

Mestl, T., Lemay, C., and Glass, L. (1996). Chaos in high-dimensional neural and gene networks. *Physica*, 98:33.

Ramsay, J. and Silverman, B. W. (1997). *Functional Data Analysis*. Springer-Verlag, New York, NY.

Yeung, K. Y., Fraley, C., Murua, A., Raftery, A. E., and Ruzzo, W. L. (2001). Model-based clustering and data transformations for gene expression data. *Bioinformatics*, 17(10):977–987.
